# Kernel Dimensionality Reduction for Supervised Learning

**Kenji Fukumizu**
Institute of Statistical
Mathematics
Tokyo 106-8569 Japan
*fukumizu@ism.ac.jp*

**Francis R. Bach**
CS Division
University of California
Berkeley, CA 94720, USA
*fbach@cs.berkeley.edu*

**Michael I. Jordan**
CS Division and Statistics
University of California
Berkeley, CA 94720, USA
*jordan@cs.berkeley.edu*

## Abstract

We propose a novel method of dimensionality reduction for supervised learning. Given a regression or classification problem in which we wish to predict a variable $Y$ from an explanatory vector $X$, we treat the problem of dimensionality reduction as that of finding a low-dimensional "effective subspace" of $X$ which retains the statistical relationship between $X$ and $Y$. We show that this problem can be formulated in terms of conditional independence. To turn this formulation into an optimization problem, we characterize the notion of conditional independence using covariance operators on reproducing kernel Hilbert spaces; this allows us to derive a contrast function for estimation of the effective subspace. Unlike many conventional methods, the proposed method requires neither assumptions on the marginal distribution of $X$, nor a parametric model of the conditional distribution of $Y$.

## 1 Introduction

Many statistical learning problems involve some form of dimensionality reduction. The goal may be one of *feature selection*, in which we aim to find linear or nonlinear combinations of the original set of variables, or one of *variable selection*, in which we wish to select a subset of variables from the original set. Motivations for such dimensionality reduction include providing a simplified explanation and visualization for a human, suppressing noise so as to make a better prediction or decision, or reducing the computational burden.

We study dimensionality reduction for supervised learning, in which the data consists of $(X, Y)$ pairs, where $X$ is an $m$-dimensional explanatory variable and $Y$ is an $\ell$-dimensional response. The variable $Y$ may be either continuous or discrete. We refer to these problems generically as "regression," which indicates our focus on the conditional probability density $p_{Y|X}(y|x)$. Thus, our framework includes classification problems, where $Y$ is discrete.

We wish to solve a problem of feature selection in which the features are linear combinations of the components of $X$. In particular, we assume that there is an $r$-dimensional subspace $S \subset \mathbb{R}^m$ such that the following equality holds for all $x$ and $y$:

$$p_{Y|X}(y|x) = p_{Y|\Pi_S X}(y|\Pi_S x), \tag{1}$$

where $\Pi_S$ is the orthogonal projection of $\mathbb{R}^m$ onto $S$. The subspace $S$ is called the *effective subspace for regression*. Based on observations of $(X, Y)$ pairs, we wish to re-

cover a matrix whose columns span $S$. We approach the problem within a *semiparametric* statistical framework—we make no assumptions regarding the conditional distribution $p_{Y|\Pi_S X}(y|\Pi_S x)$ or the distribution $p_X(x)$ of $X$. Having found an effective subspace, we may then proceed to build a parametric or nonparametric regression model on that subspace. Thus our approach is an explicit dimensionality reduction method for supervised learning that does not require any particular form of regression model; it can be used as a preprocessor for any supervised learner.

Most conventional approaches to dimensionality reduction make specific assumptions regarding the conditional distribution $p_{Y|\Pi_S X}(y|\Pi_S x)$, the marginal distribution $p_X(x)$, or both. For example, classical two-layer neural networks can be seen as attempting to estimate an effective subspace in their first layer, using a specific model for the regressor. Similar comments apply to projection pursuit regression [1] and ACE [2], which assume an additive model $E[Y|X] = g_1(\beta_1^T X) + \cdots + g_K(\beta_K^T X)$. While canonical correlation analysis (CCA) and partial least squares (PLS, [3]) can be used for dimensionality reduction in regression, they make a linearity assumption and place strong restrictions on the allowed dimensionality. The line of research that is closest to our work is sliced inverse regression (SIR, [4]) and related methods including principal Hessian directions (pHd, [5]). SIR is a semiparametric method that can find effective subspaces, but only under strong assumptions of ellipticity for the marginal distribution $p_X(x)$. pHd also places strong restrictions on $p_X(x)$. If these assumptions do not hold, there is no guarantee of finding the effective subspace.

In this paper we present a novel semiparametric method for dimensionality reduction that we refer to as *Kernel Dimensionality Reduction (KDR)*. KDR is based on a particular class of operators on reproducing kernel Hilbert spaces (RKHS, [6]). In distinction to algorithms such as the support vector machine and kernel PCA [7, 8], KDR cannot be viewed as a "kernelization" of an underlying linear algorithm. Rather, we relate dimensionality reduction to *conditional independence* of variables, and use RKHSs to provide characterizations of conditional independence and thereby design objective functions for optimization. This builds on the earlier work of [9], who used RKHSs to characterize *marginal independence* of variables. Our characterization of conditional independence is a significant extension, requiring rather different mathematical tools—the covariance operators on RKHSs that we present in Section 2.2.

## 2  Kernel method of dimensionality reduction for regression

### 2.1  Dimensionality reduction and conditional independence

The problem discussed in this paper is to find the effective subspace $S$ defined by Eq. (1), given an i.i.d. sample $\{(X_i, Y_i)\}_{i=1}^n$, sampled from the conditional probability Eq. (1) and a marginal probability $p_X$ for $X$. The crux of the problem is that we have no *a priori* knowledge of the regressor, and place no assumptions on the conditional probability $p_{Y|X}$ or the marginal probability $p_X$.

We do not address the problem of choosing the dimensionality $r$ in this paper—in practical applications of KDR any of a variety of model selection methods such as cross-validation can be reasonably considered. Rather our focus is on the problem of finding the effective subspace for a given choice of dimensionality.

The notion of effective subspace can be formulated in terms of conditional independence. Let $Q = (B, C)$ be an $m$-dimensional orthogonal matrix such that the column vectors of $B$ span the subspace $S$ (thus $B$ is $m \times r$ and $C$ is $m \times (m - r)$), and define $U = B^T X$ and $V = C^T X$. Because $Q$ is an orthogonal matrix, we have $p_X(x) = p_{U,V}(u, v)$ and $p_{X,Y}(x, y) = p_{U,V,Y}(u, v, y)$. Thus, Eq. (1) is equivalent to

$$p_{Y|U,V}(y|u, v) = p_{Y|U}(y|u). \tag{2}$$

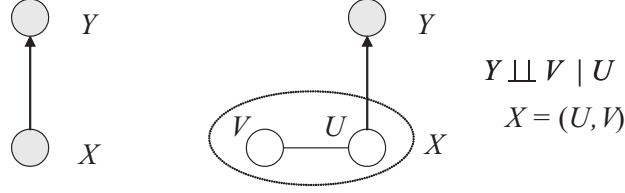

Figure 1: Graphical representation of dimensionality reduction for regression.

This shows that the effective subspace $S$ is the one which makes $Y$ and $V$ conditionally independent given $U$ (see Figure 1).

Mutual information provides another viewpoint on the equivalence between conditional independence and the effective subspace. It is well known that

$$I(Y, X) = I(Y, U) + E_U \big[ I(Y|U, V|U) \big], \qquad (3)$$

where $I(Z, W)$ is the mutual information between $Z$ and $W$. Because Eq. (1) implies $I(Y, X) = I(Y, U)$, the effective subspace $S$ is characterized as the subspace which retains the entire mutual information between $X$ and $Y$, or equivalently, such that $I(Y|U, V|U) = 0$. This is again the conditional independence of $Y$ and $V$ given $U$.

## 2.2 Covariance operators on kernel Hilbert spaces and conditional independence

We use cross-covariance operators [10] on RKHSs to characterize the conditional independence of random variables. Let $(\mathcal{H}, k)$ be a (real) reproducing kernel Hilbert space of functions on a set $\Omega$ with a positive definite kernel $k : \Omega \times \Omega \to \mathbb{R}$ and an inner product $\langle \cdot, \cdot \rangle_{\mathcal{H}}$. The most important aspect of a RKHS is the reproducing property:

$$\langle f, k(\cdot, x) \rangle_{\mathcal{H}} = f(x) \quad \text{for all } x \in \Omega \text{ and } f \in \mathcal{H}. \qquad (4)$$

In this paper we focus on the Gaussian kernel $k(x_1, x_2) = \exp\big(-\|x_1 - x_2\|^2 / 2\sigma^2\big)$.

Let $(\mathcal{H}_1, k_1)$ and $(\mathcal{H}_2, k_2)$ be RKHSs over measurable spaces $(\Omega_1, \mathcal{B}_1)$ and $(\Omega_2, \mathcal{B}_2)$, respectively, with $k_1$ and $k_2$ measurable. For a random vector $(X, Y)$ on $\Omega_1 \times \Omega_2$, the *cross-covariance operator* $\Sigma_{YX}$ from $\mathcal{H}_1$ to $\mathcal{H}_2$ is defined by the relation

$$\langle g, \Sigma_{YX} f \rangle_{\mathcal{H}_2} = E_{XY}[f(X)g(Y)] - E_X[f(X)]E_Y[g(Y)] \quad (= \text{Cov}[f(X), g(Y)]) \quad (5)$$

for all $f \in \mathcal{H}_1$ and $g \in \mathcal{H}_2$. Eq. (5) implies that the covariance of $f(X)$ and $g(Y)$ is given by the action of the linear operator $\Sigma_{YX}$ and the inner product. Under the assumption that $E_X[k_1(X, X)]$ and $E_Y[k_2(Y, Y)]$ are finite, by using Riesz's representation theorem, it is not difficult to see that a bounded operator $\Sigma_{YX}$ is uniquely defined by Eq. (5). We have $\Sigma_{YX}^* = \Sigma_{XY}$, where $A^*$ denotes the adjoint of $A$. From Eq. (5), we see that $\Sigma_{YX}$ captures all of the nonlinear correlations defined by the functions in $\mathcal{H}_X$ and $\mathcal{H}_Y$.

Cross-covariance operators provide a useful framework for discussing conditional probability and conditional independence, as shown by the following theorem and its corollary[1]:

**Theorem 1.** *Let $(\mathcal{H}_1, k_1)$ and $(\mathcal{H}_2, k_2)$ be RKHSs on measurable spaces $\Omega_1$ and $\Omega_2$, respectively, with $k_1$ and $k_2$ measurable, and $(X, Y)$ be a random vector on $\Omega_1 \times \Omega_2$. Assume that $E_X[k_1(X, X)]$ and $E_Y[k_2(Y, Y)]$ are finite, and for all $g \in \mathcal{H}_2$ the conditional expectation $E_{Y|X}[g(Y) \mid X = \cdot]$ is an element of $\mathcal{H}_1$. Then, for all $g \in \mathcal{H}_2$ we have*

$$\Sigma_{XX} E_{Y|X}[g(Y) \mid X = \cdot] = \Sigma_{XY} g. \qquad (6)$$

**Corollary 2.** *Let $\tilde{\Sigma}_{XX}^{-1}$ be the right inverse of $\Sigma_{XX}$ on $(Ker\Sigma_{XX})^{\perp}$. Under the same assumptions as Theorem 1, we have, for all $f \in (\mathrm{Ker}\Sigma_{XX})^{\perp}$ and $g \in \mathcal{H}_2$,*

$$\langle f, \tilde{\Sigma}_{XX}^{-1}\Sigma_{XY}g\rangle_{\mathcal{H}_1} = \langle f, E_{Y|X}[g(Y) \mid X = \cdot]\rangle_{\mathcal{H}_1}. \tag{7}$$

*Sketch of the proof.* $\Sigma_{XY}$ can be decomposed as $\Sigma_{XY} = \Sigma_{XX}^{1/2} V \Sigma_{YY}^{1/2}$ for a bounded operator $V$ (Theorem 1, [10]). Thus, we see $\tilde{\Sigma}_{XX}^{-1}\Sigma_{XY}$ is well-defined, because $\overline{\mathrm{Range}\Sigma_{XY}} \subset \overline{\mathrm{Range}\Sigma_{XX}} = (\mathrm{Ker}\Sigma_{XX})^{\perp}$. Then, Eq. (7) is a direct consequence of Theorem 1. $\qquad\square$

Given that $\Sigma_{XX}$ is invertible, Eq. (7) implies

$$E_{Y|X}[g(Y) \mid X = \cdot] = \Sigma_{XX}^{-1}\Sigma_{XY}g \quad \text{for all } g \in \mathcal{H}_2. \tag{8}$$

This can be understood by analogy to the conditional expectation of Gaussian random variables. If $X$ and $Y$ are Gaussian random variables, it is well-known that the conditional expectation is given by $E_{Y|X}[a^T Y \mid X = x] = x^T \Sigma_{XX}^{-1}\Sigma_{XY}a$ for an arbitrary vector $a$, where $\Sigma_{XX}$ and $\Sigma_{XY}$ are the variance-covariance matrices in the ordinary sense.

Using cross-covariance operators, we derive an objective function for characterizing conditional independence. Let $(\mathcal{H}_1, k_1)$ and $(\mathcal{H}_2, k_2)$ be RKHSs on measurable spaces $\Omega_1$ and $\Omega_2$, respectively, with $k_1$ and $k_2$ measurable, and suppose we have random variables $U \in \mathcal{H}_1$ and $Y \in \mathcal{H}_2$. We define the *conditional covariance operator* $\Sigma_{YY|U}$ on $\mathcal{H}_1$ by

$$\Sigma_{YY|U} := \Sigma_{YY} - \Sigma_{YU}\tilde{\Sigma}_{UU}^{-1}\Sigma_{UY}. \tag{9}$$

Corollary 2 easily yields the following result on the conditional covariance of variables:

**Theorem 3.** *Assume that $E_X[k_1(X, X)]$ and $E_Y[k_2(Y, Y)]$ are finite, and that $E_{Y|X}[f(Y)|X]$ is an element of $\mathcal{H}_1$ for all $f \in \mathcal{H}_2$. Then, for all $f, g \in \mathcal{H}_2$, we have*

$$\langle g, \Sigma_{YY|U}f\rangle_{\mathcal{H}_2} = E_Y[f(Y)g(Y)] - E_U\big[E_{Y|U}[f(Y)|U]E_{Y|U}[g(Y)|U]\big]$$
$$= E_U\big[\mathrm{Cov}_{Y|U}\big[f(Y), g(Y) \mid U\big]\big]. \tag{10}$$

As in the case of Eq. (8), Eqs. (9) and (10) can be viewed as the analogs of the well-known equality for Gaussian variables: $\mathrm{Cov}[a^T Y, b^T Y|U] = a^T(\Sigma_{YY} - \Sigma_{YU}\Sigma_{UU}^{-1}\Sigma_{UY})b$.

From Theorem 3, it is natural to use minimization of $\Sigma_{YY|U}$ as a basis of a method for finding the most informative $U$, which gives the least $\mathrm{Var}_{Y|U}[f(Y)|U]$. The following definition is needed to justify this intuition. Let $(\Omega, \mathcal{B})$ be a measurable space, let $(\mathcal{H}, k)$ be a RKHS over $\Omega$ with $k$ measurable and bounded, and let $\mathcal{M}$ be the set of all the probability measures on $(\Omega, \mathcal{B})$. The RKHS $\mathcal{H}$ is called *probability-determining*, if the map

$$\mathcal{M} \ni P \quad \mapsto \quad (f \mapsto E_{X\sim P}[f(X)]) \in \mathcal{H}^* \tag{11}$$

is one-to-one, where $\mathcal{H}^*$ is the dual space of $\mathcal{H}$. The following theorem can be proved using a argument similar to that used in the proof of Theorem 2 in [9].

**Theorem 4.** *For an arbitrary $\sigma > 0$, the RKHS with Gaussian kernel $k(x, y) = \exp(-\|x - y\|^2/2\sigma^2)$ on $\mathbb{R}^m$ is probability-determining.*

Recall that for two RKHSs $\mathcal{H}_1$ and $\mathcal{H}_2$ on $\Omega_1$ and $\Omega_2$, respectively, the direct product $\mathcal{H}_1 \otimes \mathcal{H}_2$ is the RKHS on $\Omega_1 \times \Omega_2$ with the kernel $k_1 k_2$ [6]. The relation between conditional independence and the conditional covariance operator is given by the following theorem:

**Theorem 5.** *Let $(\mathcal{H}_{11}, k_{11})$, $(\mathcal{H}_{12}, k_{12})$, and $(\mathcal{H}_2, k_2)$ be RKHSs on measurable spaces $\Omega_{11}$, $\Omega_{12}$, and $\Omega_2$, respectively, with continuous and bounded kernels. Let $(X, Y) = (U, V, Y)$ be a random vector on $\Omega_{11} \times \Omega_{12} \times \Omega_2$, where $X = (U, V)$, and let $\mathcal{H}_1 = \mathcal{H}_{11} \otimes \mathcal{H}_{12}$ be the direct product. It is assumed that $E_{Y|U}[g(Y)|U = \cdot] \in \mathcal{H}_{11}$ and $E_{Y|X}[g(Y)|X = \cdot] \in \mathcal{H}_1$ for all $g \in \mathcal{H}_2$. Then, we have*

$$\Sigma_{YY|U} \ge \Sigma_{YY|X}, \tag{12}$$

*where the inequality refers to the order of self-adjoint operators. If further $\mathcal{H}_2$ is probability-determining, in particular, for Gaussian kernels, we have the equivalence:*

$$\Sigma_{YY|X} = \Sigma_{YY|U} \qquad \Longleftrightarrow \qquad Y \perp\!\!\!\perp V \mid U. \tag{13}$$

*Sketch of the proof.* Taking the expectation of the well-known equality $\mathrm{Var}_{Y|U}[g(Y)|U] = E_{V|U}\big[\mathrm{Var}_{Y|U,V}[g(Y)|U,V]\big] + \mathrm{Var}_{V|U}\big[E_{Y|U,V}[g(Y)|U,V]\big]$ with respect to $U$, we obtain $E_U\big[\mathrm{Var}_{Y|U}[g(Y)|U]\big] - E_X\big[\mathrm{Var}_{Y|X}[g(Y)|X]\big] = E_U\big[\mathrm{Var}_{V|U}[E_{Y|X}[g(Y)|X]]\big] \geq 0$, which implies Eq. (12). The equality holds iff $E_{Y|X}[g(Y)|X] = E_{Y|U}[g(Y)|U]$ for a.e. $X$. Since $\mathcal{H}_2$ is probability-determining, this means $P_{Y|X} = P_{Y|U}$, that is, $Y \perp\!\!\!\perp V \mid U$. $\qquad\square$

From Theorem 5, for probability-determining kernel spaces, the effective subspace $S$ can be characterized in terms of the solution to the following minimization problem:

$$\min_S \Sigma_{YY|U}, \quad \text{subject to} \quad U = \Pi_S X. \tag{14}$$

### 2.3 Kernel generalized variance for dimensionality reduction

To derive a sampled-based objective function from Eq. (14) for a finite sample, we have to estimate the conditional covariance operator with given data, and choose a specific way to evaluate the size of self-adjoint operators. Hereafter, we consider only Gaussian kernels, which are appropriate for both continuous and discrete variables.

For the estimation of the operator, we follow the procedure in [9] (see also [11] for further details), and use the centralized Gram matrix [9, 8], which is defined as:

$$\hat{K}_Y = \left(I_n - \tfrac{1}{n}\mathbf{1}_n\mathbf{1}_n^T\right)G_Y\left(I_n - \tfrac{1}{n}\mathbf{1}_n\mathbf{1}_n^T\right), \quad \hat{K}_U = \left(I_n - \tfrac{1}{n}\mathbf{1}_n\mathbf{1}_n^T\right)G_U\left(I_n - \tfrac{1}{n}\mathbf{1}_n\mathbf{1}_n^T\right) \tag{15}$$

where $\mathbf{1}_n = (1,\ldots,1)^T$, $(G_Y)_{ij} = k_1(Y_i, Y_j)$ is the Gram matrix of the samples of $Y$, and $(G_U)_{ij} = k_2(U_i, U_j)$ is given by the projection $U_i = B^T X_i$. With a regularization constant $\varepsilon > 0$, the empirical conditional covariance matrix $\hat{\Sigma}_{YY|U}$ is then defined by

$$\hat{\Sigma}_{YY|U} := \hat{\Sigma}_{YY} - \hat{\Sigma}_{YU}\hat{\Sigma}_{UU}^{-1}\hat{\Sigma}_{UY} = (\hat{K}_Y + \varepsilon I_n)^2 - \hat{K}_Y\hat{K}_U(\hat{K}_U + \varepsilon I_n)^{-2}\hat{K}_U\hat{K}_Y. \tag{16}$$

The size of $\hat{\Sigma}_{YY|U}$ in the ordered set of positive definite matrices can be evaluated by its determinant. Although there are other choices for measuring the size of $\hat{\Sigma}_{YY|U}$, such as the trace and the largest eigenvalue, we focus on the determinant in this paper. Using the Schur decomposition, $\det(A - BC^{-1}B^T) = \det\left(\begin{smallmatrix} A & B \\ B^T & C \end{smallmatrix}\right)/\det C$, we have

$$\det \hat{\Sigma}_{YY|U} = \det \hat{\Sigma}_{[YU][YU]}/ \det \hat{\Sigma}_{UU}, \tag{17}$$

where $\hat{\Sigma}_{[YU][YU]}$ is defined by $\hat{\Sigma}_{[YU][YU]} = \left(\begin{smallmatrix} \hat{\Sigma}_{YY} & \hat{\Sigma}_{YU} \\ \hat{\Sigma}_{UY} & \hat{\Sigma}_{UU} \end{smallmatrix}\right) = \left(\begin{smallmatrix} (\hat{K}_Y + \varepsilon I_n)^2 & \hat{K}_Y\hat{K}_U \\ \hat{K}_U\hat{K}_Y & (\hat{K}_U + \varepsilon I_n)^2 \end{smallmatrix}\right)$.
We symmetrize the objective function by dividing by the constant $\det \hat{\Sigma}_{YY}$, which yields

$$\min_{B \in \mathbb{R}^{m \times r}} \frac{\det \hat{\Sigma}_{[YU][YU]}}{\det \hat{\Sigma}_{YY} \det \hat{\Sigma}_{UU}}, \quad \text{where } U = B^T X. \tag{18}$$

We refer to this minimization problem with respect to the choice of subspace $S$ or matrix $B$ as *Kernel Dimensionality Reduction (KDR)*.

Eq. (18) has been termed the "kernel generalized variance" (KGV) by Bach and Jordan [9]. They used it as a contrast function for independent component analysis, in which the goal is to *minimize* a mutual information. They showed that KGV is in fact an approximation of the mutual information among the recovered sources around the factorized distributions. In the current setting, on the other hand, our goal is to *maximize* the mutual information

|        | SIR(10) | SIR(15) | SIR(20) | SIR(25) | pHd   | KDR       |
|--------|---------|---------|---------|---------|-------|-----------|
| $R(\boldsymbol{b}_1)$ | 0.987   | 0.993   | 0.988   | 0.990   | 0.110 | **0.999** |
| $R(\boldsymbol{b}_2)$ | 0.421   | 0.705   | 0.480   | 0.526   | 0.859 | **0.984** |

Table 1: Correlation coefficients. SIR($m$) indicates the SIR method with $m$ slices.

$I(Y, U)$, and with an entirely different argument, we have shown that KGV is an appropriate objective function for the dimensionality reduction problem, and that minimizing Eq. (18) can be viewed as maximizing the mutual information $I(Y, U)$.

Given that the numerical task that must be solved in KDR is the same as the one to be solved in kernel ICA, we can import all of the computational techniques developed in [9] for minimizing KGV. In particular, the optimization routine that we use is gradient descent with a line search, where we exploit incomplete Cholesky decomposition to reduce the $n \times n$ matrices to low-rank approximations. To cope with local optima, we make use of an annealing technique, in which the scale parameter $\sigma$ for the Gaussian kernel is decreased gradually during the iterations of optimization. For a larger $\sigma$, the contrast function has fewer local optima, and the search becomes more accurate as $\sigma$ is decreased.

## 3 Experimental results

We illustrate the effectiveness of the proposed KDR method through experiments, comparing it with several conventional methods: SIR, pHd, CCA, and PLS.

The first data set is a synthesized one with 300 samples of 17 dimensional $X$ and one dimensional $Y$, which are generated by $\quad Y \sim 0.9X_1 + 0.2/(1 + X_{17}) + Z$, where $Z \sim N(0, 0.01^2)$ and $X$ follows a uniform distribution on $[0, 1]^{17}$. The effective subspace is given by $\boldsymbol{b}_1 = (1, 0, \ldots, 0)$ and $\boldsymbol{b}_2 = (0, \ldots, 0, 1)$. We compare the KDR method with SIR and pHd only—CCA and PLS cannot find a 2-dimensional subspace, because $Y$ is one-dimensional. To evaluate estimation accuracy, we use the multiple correlation coefficient $R(\boldsymbol{b}) = \max_{\boldsymbol{\beta} \in S} \boldsymbol{\beta}^T \Sigma_{XX} \boldsymbol{b} / (\boldsymbol{\beta}^T \Sigma_{XX} \boldsymbol{\beta} \cdot \boldsymbol{b}^T \Sigma_{XX} \boldsymbol{b})^{1/2}$, which is used in [4]. As shown in Table 1, KDR outperforms the others in finding the weak contribution of $\boldsymbol{b}_2$.

Next, we apply the KDR method to classification problems, for which many conventional methods of dimensionality reduction are not suitable. In particular, SIR requires the dimensionality of the effective subspace to be less than the number of classes, because SIR uses the average of $X$ in slices along the variable $Y$. CCA and PLS have a similar limitation on the dimensionality of the effective subspace. Thus we compare the result of KDR only with pHd, which is applicable to general binary classification problems.

We show the visualization capability of the dimensionality reduction methods for the *Wine* dataset from the UCI repository to see how the projection onto a low-dimensional space realizes an effective description of data. The *Wine* data consists of 178 samples with 13 variables and a label with three classes. Figure 2 shows the projection onto the 2-dimensional subspace estimated by each method. KDR separates the data into three classes most completely. We can see that the data are nonlinearly separable in the two-dimensional space.

In the third experiment, we investigate how much information on the classification is preserved in the estimated subspace. After reducing the dimensionality, we use the support vector machine (SVM) method to build a classifier in the reduced space, and compare its accuracy with an SVM trained using the full-dimensional vector $X$. We use three data sets from the UCI repository. Figure 3 shows the classification rates for the test set for subspaces of various dimensionality. We can see that KDR yields good classification even in low-dimensional subspaces, while pHd is much worse in small dimensionality. It is noteworthy that in the Ionosphere data set the classifier in dimensions 5, 10, and 20 outperforms

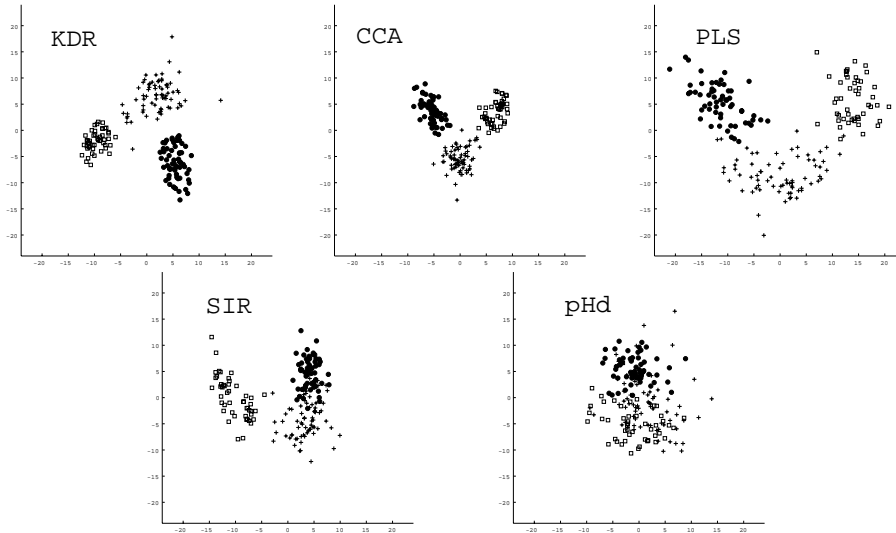

Figure 2: Projections of *Wine* data: "+", "●", and gray "□" represent the three classes.

the classifier in the full-dimensional space. This is caused by suppressing noise irrelevant to explain $Y$. These results show that KDR successfully finds an effective subspace which preserves the class information even when the dimensionality is reduced significantly.

## 4   Extension to variable selection

The KDR method can be extended to variable selection, in which a subset of given explanatory variables $\{X_1, \ldots, X_m\}$ is selected. Extension of the KGV objective function to variable selection is straightforward. We have only to compare the KGV values for all the subspaces spanned by combinations of a fixed number of selected variables. We of course do not avoid the combinatorial problem of variable selection; the total number of combinations may be intractably large for a large number of explanatory variables $m$, and greedy or random search procedures are needed.

We first apply this kernel method to the *Boston Housing* data (506 samples with 13 dimensional $X$), which has been used as a typical example of variable selection. We select four variables that attain the smallest KGV value among all the combinations. The selected variables are exactly the same as the ones selected by ACE [2]. Next, we apply the method to the *leukemia* microarray data of 7129 dimensions ([12]). We select 50 effective genes to classify two types of leukemia using 38 training samples. For optimization of the KGV value, we use a greedy algorithm, in which new variables are selected one by one, and subsequently a variant of genetic algorithm is used. Half of the 50 genes accord with 50 genes selected by [12]. With the genes selected by our method, the same classifier as that used in [12] classifies correctly 32 of the 34 test samples, for which, with their 50 genes, Golub *et al.* ([12]) report a result of classifying 29 of the 34 samples correctly.

## 5   Conclusion

We have presented KDR, a novel method of dimensionality reduction for supervised learning. One of the striking properties of this method is its generality. We do not place any strong assumptions on either the conditional or the marginal distribution, in distinction to

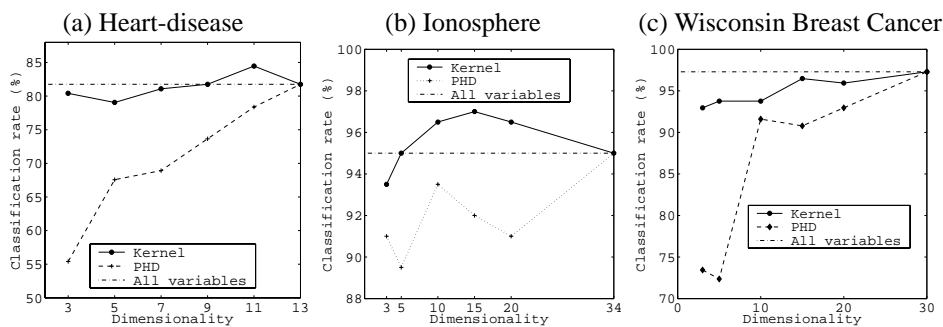

Figure 3: Classification accuracy of the SVM for test data after dimensionality reduction.

essentially all existing methods for dimensionality reduction in regression, including SIR, pHd, CCA, and PPR. We have demonstrating promising empirical performance of KDR, showing its practical utility in data visualization and feature selection for prediction. We have also discussed an extension of KDR method to variable selection.

The theoretical basis of KDR lies in the nonparametric characterization of conditional independence that we have presented in this paper. Extending earlier work on the kernel-based characterization of marginal independence [9], we have shown that conditional independence can be characterized in terms of covariance operators on a kernel Hilbert space. While our focus has been on the problem of dimensionality reduction, it is also worth noting that there are many possible other applications of this result. In particular, conditional independence plays an important role in the structural definition of graphical models, and our result may have implications for model selection and inference in graphical models.

## Footnotes

[1]Full proofs of all theorems can be found in [11].

# References

[1] Friedman, J.H. and Stuetzle, W. Projection pursuit regression. *J. Amer. Stat. Assoc.*, 76:817–823, 1981.

[2] Breiman, L. and Friedman, J.H. Estimating optimal transformations for multiple regression and correlation. *J. Amer. Stat. Assoc.*, 80:580–598, 1985.

[3] Wold, H. Partial least squares. in S. Kotz and N.L. Johnson (Eds.), *Encyclopedia of Statistical Sciences, Vol. 6*, Wiley, New York. pp.581–591. 1985.

[4] Li, K.-C. Sliced inverse regression for dimension reduction (with discussion). *J. Amer. Stat. Assoc.*, 86:316–342, 1991.

[5] Li, K.-C. On principal Hessian directions for data visualization and dimension reduction: Another application of Stein's lemma. *J. Amer. Stat. Assoc.*, 87:1025–1039, 1992.

[6] Aronszajn, N. Theory of reproducing kernels. *Trans. Amer. Math. Soc.*, 69(3):337–404, 1950.

[7] Schölkopf, B., Burges, C.J.C., and Smola, A. (eds.) *Advances in Kernel Methods: Support Vector Learning*. MIT Press. 1999.

[8] Schölkopf, B., Smola, A and Müller, K.-R. Nonlinear component analysis as a kernel eigenvalue problem. *Neural Computation*, 10:1299–1319, 1998.

[9] Bach, F.R. and Jordan, M.I. Kernel independent component analysis. *JMLR*, 3:1–48, 2002.

[10] Baker, C.R. Joint measures and cross-covariance operators. *Trans. Amer. Math. Soc.*, 186:273–289, 1973.

[11] Fukumizu, K., Bach, F.R. and Jordan, M.I. Dimensionality reduction for supervised learning with reproducing kernel Hilbert spaces. *JMLR*, 5:73–99, 2004.

[12] Golub T.R. *et al.* Molecular classification of cancer: Class discovery and class prediction by gene expression monitoring. *Science*, 286:531–537, 1999.
